# Computational Equivalence of Fixed Points and No Regret Algorithms, and Convergence to Equilibria

**Elad Hazan**
IBM Almaden Research Center
650 Harry Road
San Jose, CA 95120
hazan@us.ibm.com

**Satyen Kale**
Computer Science Department,
Princeton University
35 Olden St.
Princeton, NJ 08540
satyen@cs.princeton.edu

## Abstract

We study the relation between notions of game-theoretic equilibria which are based on stability under a set of deviations, and empirical equilibria which are reached by rational players. Rational players are modeled by players using no regret algorithms, which guarantee that their payoff in the long run is close to the maximum they could hope to achieve by consistently deviating from the algorithm's suggested action.

We show that for a given set of deviations over the strategy set of a player, it is possible to efficiently approximate fixed points of a given deviation if and only if there exist efficient no regret algorithms resistant to the deviations. Further, we show that if all players use a no regret algorithm, then the empirical distribution of their plays converges to an equilibrium.

## 1 Introduction

We consider a setting where a number of agents need to repeatedly make decisions in the face of uncertainty. In each round, the agent obtains a payoff based on the decision she chose. Each agent would like to be able to maximize her payoff. While this might seem like a natural objective, it may be impossible to achieve without placing restrictions on the kind of payoffs that can arise. For instance, if the payoffs were adversarially chosen, then the agent's task would become essentially hopeless.

In such a situation, one way for the agent to cope with the uncertainty is to aim for a relative benchmark rather an absolute one. The notion of *regret minimization* captures this intuition. We imagine that the agent has a choice of several well-defined ways to change her decision, and now the agent aims to maximize her payoff relative to what she could have obtained had she changed her decisions in a consistent manner. As an example of what we mean by consistent changes, a possible objective could be to maximize her payoff relative to the most she could have achieved by choosing some fixed decision in all the rounds. The difference between these payoffs is known as *external regret* in the game theory literature. Another notion is that of *internal regret*, which arises when the possible ways to change are the ones that switch from some decision $i$ to another, $j$, whenever the agent chose decision $i$, leaving all other decisions unchanged.

A learning algorithm for an agent is said to have no regret with respect to an associated set of decision modifiers (also called deviations) $\Phi$ if the average payoff of an agent using the algorithm converges to the largest average payoff she would have achieved had she changed her decisions using a fixed decision modifier in all the rounds. Based on what set of decision modifiers are under consideration, various no regret algorithms are known (for e.g. Hannan [10] gave algorithms to minimize external regret, and Hart and Mas-Collel [11] give algorithms to minimize internal regret).

The reason no regret algorithms are so appealing, apart from the fact that they model rational behavior of agents in the face of uncertainty, is that in various cases it can be shown that using no regret algorithms guides the overall play towards a game theoretic equilibrium. For example, Freund and Schapire [7] show that in a zero-sum game, if all agents use a no external regret algorithm, then the empirical distribution of the play converges to the set of minimax equilibria. Similarly, Hart and Mas-Collel [11] show that if all agents use a no internal regret algorithm, then the empirical distribution of the play converges to the set of correlated equilibria.

In general, given a set of decision modifiers $\Phi$, we can define a notion of game theoretic equilibrium that is based on the property of being stable under deviations specified by $\Phi$. This is a joint distribution on the agents' decisions that ensures that the expected payoff to any agent is no less than the most she could achieve if she decided to unilaterally (and consistently) decided to deviate from her suggested action using any decision modifier in $\Phi$. One can then show that if all agents use a $\Phi$-no regret algorithm, then the empirical distribution of the play converges to the set of $\Phi$-equilibria.

This brings us to the question of whether it is possible to design no regret algorithms for various sets of decision modifiers $\Phi$. In this paper, we design algorithms which achieve no regret with respect to $\Phi$ for a very general setting of arbitrary convex compact decision spaces, arbitrary concave payoff functions, and arbitrary continuous decision modifiers. Our method works as long as it is possible to compute approximate fixed points for (convex combinations) of decision modifiers in $\Phi$. Our algorithms are based on a connection to the framework of Online Convex Optimization (see, e.g. [18]) and we show how to apply known learning algorithms to obtain $\Phi$-no regret algorithms. The generality of our connection allows us to use various sophisticated Online Convex Optimization algorithms which can exploit various structural properties of the utility functions and guarantee a faster rate of convergence to the equilibrium.

Previous work by Greenwald and Jafari [9] gave algorithms for the case when the decision space is the simplex of probability distributions over the agents' decisions, the payoff functions are linear, and the decision modifiers are also linear. Their algorithm, based on the work of Hart and Mas-Collel [11], uses a version of Blackwell's Approachability Theorem, and also needs to computes fixed points of the decision modifiers. Since these modifiers are linear, it is possible to compute fixed points for them by computing the stationary distribution of an appropriate stochastic matrix (say, by computing its top eigenvector).

Computing Brouwer fixed points of continuous functions is in general a very hard problem (it is PPAD-complete, as shown by Papadimitriou [15]). Fixed points are ubiquitous in game theory. Most common notions of equilibria in game theory are defined as the set of fixed points of a certain mapping. For example, Nash Equilibria (NE) are the set of fixed points of the best response mapping (appropriately defined to avoid ambiguity). The fact that Brouwer fixed points are hard to compute in general is no reason why computing specific fixed points should be hard (for instance, as mentioned earlier, computing fixed points of linear functions is easy via eigenvector computations). More specifically, could it be the case that the NE, being a fixed point of some well-specified mapping, is easy to compute? These hopes were dashed by the work of [6, 3] who showed that computing NE is as computationally difficult as finding fixed points in a general mapping: they show that computing NE in a two-player game is PPAD-complete. Further work showed that even computing an approximate NE is PPAD-complete [4].

Since our algorithms (and all previous ones as well) depend on computing (approximate) fixed points of various decision modifiers, the above discussion leads us to question whether this is necessary. We show in this paper that indeed it is: a $\Phi$-no-regret algorithm can be efficiently used to compute approximate fixed points of any convex combination of decision modifiers. This establishes an equivalence theorem, which is the main contribution of this paper: there exist efficient $\Phi$-no-regret algorithms if and only it is possible to efficiently compute fixed points of convex combinations of decision modifiers in $\Phi$. This equivalence theorem allows us to translate complexity theoretic lower bounds on computing fixed points to designing no regret algorithms. For instance, a Nash equilibrium can be obtained by applying Brouwer's fixed point theorem to an appropriately defined continuous mapping from the compact convex set of pairs of the players' mixed strategies to itself. Thus, if $\Phi$ contains this mapping, then it is PPAD-hard to design $\Phi$-no-regret algorithms.

It was recently brought to our attention that Stolz and Lugosi [17], building on the work of Hart and Schmeidler [12], have also considered $\Phi$-no-regret algorithms. They also show how to design them

from fixed-point oracles, and proved convergence to equilibria under even more general conditions than we consider. Gordon, Greenwald, Marks, and Zinkevich [8] have also considered similar notions of regret and showed convergence to equilibria, in the special case when the deviations in $\Phi$ can be represented as the composition of a fixed embedding into a higher dimensional space and an adjustable linear transformation. The focus of our results is on the computational aspect of such reductions, and the equivalence of fixed-points computation and no-regret algorithms.

## 2 Preliminaries

### 2.1 Games and Equilibria

We consider the following kinds of games. First, the set of strategies for the players of the game is a convex compact set. Second, the utility functions for the players are concave over their strategy sets. To avoid cumbersome notation, we restrict ourselves to two player games, although all of our results naturally extend to multi-player games.

Formally, for $i = 1, 2$, player $i$ plays points from a convex compact set $K_i \subseteq \mathbb{R}^{n_i}$. Her payoff is given by function $u_i : K_1 \times K_2 \to \mathbb{R}$, i.e. if $\mathbf{x}_1, \mathbf{x}_2$ is the pair of strategies played by the two players, then the payoff to player $i$ is given by $u_i(\mathbf{x}_1, \mathbf{x}_2)$. We assume that $u_1$ is a concave function of $\mathbf{x}_1$ for any fixed $\mathbf{x}_2$, and similarly $u_2$ is a concave function of $\mathbf{x}_2$ for any fixed $\mathbf{x}_1$.

We now define a notion of game theoretic equilibrium based on the property of being stable with respect to consistent deviations. By this, we mean an online game-playing strategy for the players that will guarantee that neither stands to gain if they decided to unilaterally, and consistently, deviate from their suggested moves.

To model this, assume that each player $i$ has a set of possible deviations $\Phi_i$ which is a finite[1] set of continuous mappings $\phi_i : K_i \to K_i$. Let $\Phi = (\Phi_1, \Phi_2)$. Let $\Psi$ be a joint distribution on $K_1 \times K_2$. If it is the case that for any deviation $\phi_1 \in \Phi_1$, player 1's expected payoff obtained by sampling $\mathbf{x}_1$ using $\Psi$ is always larger than her expected payoff obtained by deviating to $\phi_1(\mathbf{x}_1)$, then we call $\Psi$ *stable under deviations in* $\Phi_1$. The distribution $\Psi$ is said to be a $\Phi$-equilibrium if $\Psi$ is stable under deviations in $\Phi_1$ and $\Phi_2$. A similar definition appears in [12] and [17].

**Definition 1** ($\Phi$-equilibrium). *A joint distribution $\Psi$ over $K_1 \times K_2$ is called a $\Phi$-equilibrium if the following holds, for any $\phi_1 \in \Phi_1$, and for any $\phi_2 \in \Phi_2$:*

$$\int u_1(\mathbf{x}_1, \mathbf{x}_2)\Psi(\mathbf{x}_1, \mathbf{x}_2) \geq \int u_1(\phi_1(\mathbf{x}_1), \mathbf{x}_2)\Psi(\mathbf{x}_1, \mathbf{x}_2)$$

$$\int u_2(\mathbf{x}_1, \mathbf{x}_2)\Psi(\mathbf{x}_1, \mathbf{x}_2) \geq \int u_2(\mathbf{x}_1, \phi_2(\mathbf{x}_2))\Psi(\mathbf{x}_1, \mathbf{x}_2)$$

*We say that $\Psi$ is a $\varepsilon$-approximate $\Phi$-equilibrium if the inequalities above are satisfied up to an additive error of $\varepsilon$.*

Intuitively, we imagine a repeated game between the two players, where at equilibrium, the players' moves are correlated by a signal, which could be the past history of the play, and various external factors. This signal samples a pair of moves from an equilibrium joint distribution over all pairs of moves, and suggests to each player individually only the move she is supposed to play. If no player stands to gain if she unilaterally, but consistently, used a deviation from her suggested move, then the distribution of the correlating signal is stable under the set of deviations, and is hence an equilibrium.

**Example 1: Correlated Equilibria.** A standard 2-player game is obtained when the $K_i$ are the simplices of distributions over some base sets of actions $A_i$ and the utility functions $\mathbf{u}_i$ are bilinear in $\mathbf{x}_1, \mathbf{x}_2$. If the sets $\Phi_i$ consist of the maps $\phi_{a,b} : K_i \to K_i$ for every pair $a, b \in A_i$ defined as

$$\phi_{a,b}(\mathbf{x})[c] = \begin{cases} 0 & \text{if } c = a \\ \mathbf{x}_a + \mathbf{x}_b & \text{if } c = b \\ \mathbf{x}_c & \text{otherwise} \end{cases} \tag{1}$$

then it can be shown that any $\Phi$-equilibrium can be equivalently viewed as a correlated equilibrium of the game, and *vice-versa*.

**Example 2: The Stock Market game.** Consider the following setting: there are two investors (the generalization to many investors is straightforward), who invest their wealth in $n$ stocks. In each period, they choose portfolios $\mathbf{x}_1$ and $\mathbf{x}_2$ over the $n$ stocks, and observe the stock returns. We model the stock returns as a function $r$ of the portfolios $\mathbf{x}_1, \mathbf{x}_2$ chosen by the investors, and it maps the portfolios to the vector of stock returns. We make the assumption that each player has a small influence on the market, and thus the function $r$ is insensitive to the small perturbations in the input.

The wealth gain for each investor $i$ is $r(\mathbf{x}_1, \mathbf{x}_2) \cdot \mathbf{x}_i$. The standard way to measure performance of an investment strategy is the logarithmic growth rate, viz. $\log(r(\mathbf{x}_1, \mathbf{x}_2) \cdot \mathbf{x}_i)$. We can now define the utility functions as $u_i(\mathbf{x}_1, \mathbf{x}_2) = \log(r(\mathbf{x}_1, \mathbf{x}_2) \cdot \mathbf{x}_i)$. Intuitively, this game models the setting in which the market prices are affected by the investments of the players.

A natural goal for a good investment strategy would be to compare the wealth gain to that of the best fixed portfolio, i.e. $\Phi_i$ is the set of all constant maps. This was considered by Cover in his Universal Portfolio Framework [5]. Another possible goal would be to compare the wealth gained to that achievable by modifying the portfolios using the $\phi_{a,b}$ maps above, as considered by [16]. In Section 3, we show that the stock market game admits algorithms that converge to an $\varepsilon$-equilibrium in $O(\frac{1}{\varepsilon} \log \frac{1}{\varepsilon})$ rounds, whereas all previous algorithms need $O(\frac{1}{\varepsilon^2})$ rounds.

## 2.2 No regret algorithms

The online learning framework we consider is called *online convex optimization* [18], in which there is a fixed convex compact feasible set $K \subset \mathbb{R}^n$ and an *arbitrary, unknown* sequence of concave payoff functions $f^{(1)}, f^{(2)}, \ldots : K \to \mathbb{R}$. The decision maker must make a sequence of decisions, where the $t^{\text{th}}$ decision is a selection of a point $\mathbf{x}^{(t)} \in K$ and obtains a payoff of $f^{(t)}(\mathbf{x}^{(t)})$ on period $t$. The decision maker can only use the previous points $\mathbf{x}^{(1)}, \ldots, \mathbf{x}^{(t-1)}$, and the previous payoff functions $f^{(1)}, \ldots, f^{(t-1)}$ to choose the point $\mathbf{x}^{(t)}$.

The performance measure we use to evaluate online algorithms is regret, defined as follows. The decision maker has a finite set of $N$ decision modifiers $\Phi$ which, as before, is a set of continuous mappings from $K \to K$. Then the regret for not using some deviation $\phi \in \Phi$ is the excess payoff the decision maker could have obtained if she had changed her points in each round by applying $\phi$.

**Definition 2** ($\Phi$-Regret). *Let $\Phi$ be a set of continuous functions from $K \to K$. Given a set of $T$ concave utility functions $f_1, ..., f_T$, define the $\Phi$-regret as*

$$\text{Regret}_\Phi(T) \;=\; \max_{\phi \in \Phi} \sum_{t=1}^{T} f^{(t)}(\phi(\mathbf{x}^{(t)})) - \sum_{t=1}^{T} f^{(t)}(\mathbf{x}^{(t)}).$$

Two specific examples of $\Phi$-regret deserve mention. The first one is "external regret", which is defined when $\Phi$ is the set of all constant mappings from $K$ to itself. The second one is "internal regret", which is defined when $K$ is the simplex of distributions over some base set of actions $A$, and $\Phi$ is the set of the $\phi_{a,b}$ functions (defined in (1)) for all pairs $a, b \in A$.

A desirable property of an algorithm for Online Convex Optimization is *Hannan consistency*: the regret, as a function of the number of rounds $T$, is sublinear. This implies that the average per iteration payoff of the algorithm converges to the average payoff of a clairvoyant algorithm that uses the best deviation in hindsight to change the point in every round. For the purpose of this paper, we require a slightly stronger property for an algorithm, viz. that the regret is polynomially sublinear as a function of $T$.

**Definition 3** (No $\Phi$-regret algorithm). *A no $\Phi$-regret algorithm is one which, given any sequence of concave payoff functions $f^{(1)}, f^{(2)}, \ldots$, generates a sequence of points $\mathbf{x}^{(1)}, \mathbf{x}^{(2)}, \ldots \in K$ such that for all $T = 1, 2, \ldots$, $\text{Regret}_\Phi(T) = O(T^{1-c})$ for some constant $c > 0$. Such an algorithm will be called efficient if it computes $\mathbf{x}^{(t)}$ in $poly(n, N, t, L)$ time.*

In the above definition, $L$ is a description length parameter for $K$, defined appropriately depending on how the set $K$ is represented. For instance, if $K$ is the $n$-dimensional probability simplex, then

$L = n$. If $K$ is specified by means of a separation oracle and inner and outer radii $r$ and $R$, then $L = \log(R/r)$, and we allow $\text{poly}(n, N, t, L)$ calls to the separation oracle in each iteration.

The relatively new framework of Online Convex Optimization (OCO) has received much attention recently in the machine learning community. Our no $\Phi$-regret algorithms can use any of wide variety of algorithms for OCO. In this paper, we will use Exponentiated Gradient (EG) algorithm ([14], [1]), which has the following (external) regret bound:

**Theorem 1.** *Let the domain $K$ be the simplex of distributions over a base set of size $n$. Let $G_\infty$ be an upper bound on the $\mathcal{L}_\infty$ norm of the gradients of the payoff functions, i.e. $G_\infty \geq \sup_{\mathbf{x} \in K} \|\nabla f^{(t)}(\mathbf{x})\|_\infty$. Then the EG algorithm generates points $\mathbf{x}^{(1)}, \ldots, \mathbf{x}^{(T)}$ such that*

$$\max_{\mathbf{x} \in K} \sum_{t=1}^{T} f^{(t)}(\mathbf{x}) - \sum_{t=1}^{T} f^{(t)}(\mathbf{x}^{(t)}) \leq O(G_\infty \sqrt{\log(n)T})$$

If the utility functions are strictly concave rather than linear, even stronger regret bounds, which depend on $\log(T)$ rather than $\sqrt{T}$, are known [13].

While most of the literature on online convex optimization focuses on external regret, it was observed that *any* Online Convex Optimization algorithm for external regret can be converted to an internal regret algorithm (for example, see [2], [16]).

## 2.3 Fixed Points

As mentioned in the introduction, our no regret algorithms depend on computing fixed points of the relevant mappings. For a given set of deviations $\Phi$, denote by $\text{CH}(\Phi)$ the set of all convex combinations of deviations in $\Phi$, i.e.

$$\text{CH}(\Phi) = \left\{ \sum_{\phi \in \Phi} \alpha_\phi \phi : \ \alpha_\phi \geq 0 \text{ and } \sum_{\phi \in \Phi} \alpha_\phi = 1 \right\}.$$

Since each map $\phi \in \text{CH}(\Phi)$ is a continuous function from $K \to K$, and $K$ is a convex compact domain, by Brouwer's fixed theorem, $\phi$ has a fixed point in $K$, i.e. there exists a point $\mathbf{x} \in K$ such that $\phi(\mathbf{x}) = \mathbf{x}$. We consider algorithms which approximate fixed points for a given map in the following sense.

**Definition 4** (FPTAS for fixed points of deviations)**.** *Let $\Phi$ be a set of $N$ continuous functions from $K \to K$. A fully polynomial time approximation scheme (FPTAS) for fixed points of $\Phi$ is an algorithm, which, given any function $\phi \in \text{CH}(\Phi)$ and an error parameter $\varepsilon > 0$, computes a point $\mathbf{x} \in K$ such that $\|\phi(\mathbf{x}) - \mathbf{x}\| \leq \varepsilon$ in $\text{poly}(n, N, L, \frac{1}{\varepsilon})$ time.*

## 3 Convergence of no $\Phi$-regret algorithms to $\Phi$-equilibria

In this section we prove that if the players use no $\Phi$-regret algorithms, then the empirical distribution of the moves converges to a $\Phi$-equilibrium. [11] shows that if players use no internal regret algorithms, then the empirical distribution of the moves converges to a correlated equilibrium. This was generalized by [9] to any set of linear transformations $\Phi$. The more general setting of this paper also follows easily from the definitions. A similar theorem was also proved in [17].

The advantage of this general setting is that the connection to online convex optimization allows for faster rates of convergence using recent online learning techniques. We give an example of a natural game theoretic setting with faster convergence rate below.

**Theorem 2.** *If each player $i$ chooses moves using a no $\Phi_i$-regret algorithms, then the empirical game distribution of the players' moves converges to a $\Phi$-equilibrium. Further, an $\varepsilon$-approximate $\Phi$-equilibrium is reached after $T$ iterations for the first $T$ which satisfies $\frac{1}{T} Regret_\Phi(T) \leq \varepsilon$.*

*Proof.* Consider the first player. In each game iteration $t$, let $(\mathbf{x}_1^{(t)}, \mathbf{x}_2^{(t)})$ be the pair of moves played by the two players. From player 1's point of view, the payoff function she obtains, $f^{(t)}$, is the following:

$$\forall \mathbf{x} \in K_1 : \quad f^{(t)}(\mathbf{x}) \triangleq u_1(\mathbf{x}, \mathbf{x}_2^{(t)}).$$

Note that this function is concave by assumption. Then we have, by definition 3,

$$\text{Regret}_{\Phi_1}(T) = \max_{\phi \in \Phi} \sum_t f^{(t)}(\phi(\mathbf{x}_1^{(t)})) - \sum_t f^{(t)}(\mathbf{x}_1^{(t)}).$$

Rewriting this in terms of the original utility function, and scaling by the number of iterations we get

$$\frac{1}{T} \sum_{t=1}^{T} u_1(\mathbf{x}_1^{(t)}, \mathbf{x}_2^{(t)}) \geq \frac{1}{T} \sum_{t=1}^{T} u_1(\phi(\mathbf{x}_1^{(t)}), \mathbf{x}_2^{(t)}) - \frac{1}{T}\text{Regret}_{\Phi_1}(T).$$

Denote by $\Psi^{(T)}$ the empirical distribution of the played strategies till iteration $T$, i.e. the distribution which puts a probability mass of $\frac{1}{T}$ on all pairs $(\mathbf{x}_1^{(t)}, \mathbf{x}_2^{(t)})$ for $t = 1, 2, \ldots, T$. Then, the above inequality can be rewritten as

$$\int u_1(\mathbf{x}_1, \mathbf{x}_2)\Psi^{(T)}(\mathbf{x}_1, \mathbf{x}_2) \geq \int u_1(\phi(\mathbf{x}_1), \mathbf{x}_2)\Psi^{(T)}(\mathbf{x}_1, \mathbf{x}_2) - \frac{1}{T}\text{Regret}_{\Phi_1}(T).$$

A similar inequality holds for player 2 as well. Now assume that both players use no regret algorithms, which ensure that $\text{Regret}_{\Phi_i}(T) \leq O(T^{1-c})$ for some constant $c > 0$. Hence as $T \to \infty$, we have $\frac{1}{T}\text{Regret}_{\Phi_i}(T) \to 0$. Thus $\Psi^{(T)}$ converges to a $\Phi$-equilibrium. Also, $\Psi^{(T)}$ is a $\varepsilon$-approximate equilibrium as soon as $T$ is large enough so that $\frac{1}{T}\text{Regret}_{\Phi_1}(T)$ and $\frac{1}{T}\text{Regret}_{\Phi_2}(T)$ are less than $\varepsilon$, i.e. $T \geq \Omega(\frac{1}{\varepsilon^{1/c}})$. $\qquad\square$

A corollary of Theorem 2 is that we can obtain faster rates of convergence using recent online learning techniques, when the payoff functions are non-linear. This is natural in many situations, since risk aversion is associated with the concavity of utility functions.

**Corollary 3.** *For the stock market game as defined in section 2.1, there exists no regret algorithms which guarantee convergence to an $\varepsilon$-equilibrium in $O(\frac{1}{\varepsilon} \log \frac{1}{\varepsilon})$ iterations.*

*Proof sketch.* The utility functions observed by the investor $i$ in the stock market game are of the form $u_i(\mathbf{x}_1, \mathbf{x}_2) = \log(r(\mathbf{x}_1, \mathbf{x}_2) \cdot \mathbf{x}_i)$. This logarithmic utility function is exp-concave, by the assumption on the insensitivity of the function $r$ to small perturbations in the input. Thus the online algorithm of [5], or the more efficient algorithms of [13] can be applied. In the full version of this paper, we show that Lemma 6 can be modified to obtain algorithms with $\text{Regret}_{\Phi_i}(T) = O(\log T)$. By the Theorem 2 above, the investors reach $\varepsilon$-equilibrium in $O(\frac{1}{\varepsilon} \log \frac{1}{\varepsilon})$ iterations. $\qquad\square$

# 4 Computational Equivalence of Fixed Points and No Regret algorithms

In this section we prove our main result on the computational equivalence of computing fixed points and designing no regret algorithms. By the result of the previous section, players using no regret algorithms converge to equilibria.

We assume that the payoff functions $f^{(t)}$ are scaled so that the ($\mathcal{L}_2$) norm of their gradients is bounded by 1, i.e. $\|\nabla f^{(t)}\| \leq 1$. Our main theorem is the following:

**Theorem 4.** *Let $\Phi$ be a given finite set of deviations. Then there is a FPTAS for fixed points of $\Phi$ if and only if there exists an efficient no $\Phi$-regret algorithm.*

The first direction of the theorem is proved by designing utility functions for which the no regret property will imply convergence to an approximate fixed point of the corresponding transformations. The proof crucially depends on the fact that no regret algorithms have the stringent requirement that their worst case regret, against arbitrary adversarially chosen payoff functions, is sublinear as a function of the number of the rounds.

**Lemma 5.** *If there exists a no $\Phi$-regret algorithm then there exists an FPTAS for fixed points of $\Phi$.*

*Proof.* Let $\phi_0 \in \text{CH}(\Phi)$ be a given mapping whose fixed point we wish to compute. Let $\varepsilon$ be a given error parameter.

At iteration $t$, let $\mathbf{x}^{(t)}$ be the point chosen by $\mathcal{A}$. If $\|\phi_0(\mathbf{x}^{(t)}) - \mathbf{x}^{(t)}\| \leq \varepsilon$, we can stop, because we have found an approximate fixed point. Else, supply $\mathcal{A}$ with the following payoff function:

$$f^{(t)}(\mathbf{x}) \triangleq \frac{(\phi_0(\mathbf{x}^{(t)}) - \mathbf{x}^{(t)})^\top}{\|\phi_0(\mathbf{x}^{(t)}) - \mathbf{x}^{(t)}\|}(\mathbf{x} - \mathbf{x}^{(t)})$$

This is a linear function, with $\|\nabla f^{(t)}(\mathbf{x})\| = 1$. Also, $f^{(t)}(\mathbf{x}^{(t)}) = 0$, and $f^{(t)}(\phi_0(\mathbf{x}^{(t)})) = \|\phi_0(\mathbf{x}^{(t)}) - \mathbf{x}^{(t)}\| \geq \varepsilon$. After $T$ iterations, since $\phi_0$ is a convex combination of functions in $\Phi$, and since all the $f^{(t)}$ are linear functions, we have

$$\max_{\phi \in \Phi} \sum_{t=1}^{T} f^{(t)}(\phi(\mathbf{x}^{(t)})) \geq \sum_{t=1}^{T} f^{(t)}(\phi_0(\mathbf{x}^{(t)})) \geq \varepsilon T.$$

Thus,

$$\text{Regret}_\Phi(T) = \max_{\phi \in \Phi} \sum_t f^{(t)}(\phi(\mathbf{x}^{(t)})) - \sum_t f^{(t)}(\mathbf{x}^{(t)}) \geq \varepsilon T. \tag{2}$$

Since $\mathcal{A}$ is a no-regret algorithm, assume that $\mathcal{A}$ ensures that $\text{Regret}_\Phi(T) = O(T^{1-c})$ for some constant $c > 0$. Thus, when $T = \Omega(\frac{1}{\varepsilon^{1/c}})$ the lower bound (2) on the regret cannot hold unless we have already found an $\varepsilon$-approximate fixed point of $\phi_0$. $\qquad\square$

The second direction is on the lines of the algorithms of [2] and [16] which use fixed point computations to obtain no internal regret algorithms.

**Lemma 6.** *If there is an FPTAS for fixed points of $\Phi$, then there is an efficient no $\Phi$-regret algorithm. In fact, the algorithm guarantees that* $\text{Regret}_\Phi(T) = O(\sqrt{T})$. [2]

*Proof.* We reduce the given OCO problem to an "inner" OCO problem. The "outer" OCO problem is the original one. We use a no external regret algorithm for the inner OCO problem to generate points in $K$ for the outer one, and use the payoff functions obtained in the outer OCO problem to generate appropriate payoff functions for the inner one.

Let $\Phi = \{\phi_1, \phi_2, \ldots, \phi_N\}$. The domain for the inner OCO problem is the simplex of all distributions on $\Phi$, denoted $\Delta_N$. For a distribution $\alpha \in \Delta_N$, let $\alpha_i$ be the probability measure assigned to $\phi_i$ in the distribution $\alpha$. There is a natural mapping from $\Delta_N \to \text{CH}(\Phi)$: for any $\alpha \in \Delta_N$, denote by $\phi_\alpha$ the function $\sum_{i=1}^{N} \alpha_i \phi_i \in \text{CH}(\Phi)$.

Let $\mathbf{x}^{(t)} \in K$ be the point used in the outer OCO problem in the $t^{\text{th}}$ round, and let $f^{(t)}$ be the obtained payoff function. Then the payoff functions for the inner OCO problem is the function $g^{(t)} : \Delta_N \to \mathbb{R}$ defined as follows:

$$\forall \alpha \in \Delta_N: \quad g^{(t)}(\alpha) \triangleq f^{(t)}(\phi_\alpha(\mathbf{x}^{(t)})).$$

We now apply the Exponentiated Gradient (EG) algorithm (see Section 2.2) to the inner OCO problem. To analyze the algorithm, we bound $\|\nabla g^{(t)}\|_\infty$ as follows. Let $\mathbf{x}_0$ be an arbitrary point in $K$. We can rewrite $g^{(t)}$ as $g^{(t)}(\alpha) = f^{(t)}(\mathbf{x}_0 + \sum_i \alpha_i(\phi_i(\mathbf{x}^{(t)}) - \mathbf{x}_0))$, because $\sum_i \alpha_i = 1$. Then, $\nabla g^{(t)} = \mathbf{X}^{(t)} \nabla f^{(t)}(\phi_\alpha(\mathbf{x}^{(t)}))$, where $\mathbf{X}^{(t)}$ is an $N \times n$ matrix whose $i^{\text{th}}$ row is $(\phi_i(\mathbf{x}^{(t)}) - \mathbf{x}_0)^\top$. Thus,

$$\|\nabla g^{(t)}\|_\infty = \max_i |(\phi_i(\mathbf{x}^{(t)}) - \mathbf{x}_0)^\top \nabla f^{(t)}(\phi_\alpha(\mathbf{x}^{(t)}))| \leq \|\phi_i(\mathbf{x}^{(t)}) - \mathbf{x}_0\| \|\nabla f^{(t)}(\phi_\alpha(\mathbf{x}^{(t)}))\| \leq 1.$$

The last inequality follows because we assumed that the diameter of $K$ is bounded by 1, and the norm of the gradient of $f^{(t)}$ is also bounded by 1.

Let $\alpha^{(t)}$ be the distribution on $\Phi$ produced by the EG algorithm at time $t$. Now, the point $\mathbf{x}^{(t)}$ is computed by running the FPTAS for computing an $\frac{1}{\sqrt{t}}$-approximate fixed point of the function $\phi_{\alpha^{(t)}}$, i.e. we have $\|\phi_{\alpha^{(t)}}(\mathbf{x}^{(t)}) - \mathbf{x}^{(t)}\| \leq \frac{1}{\sqrt{t}}$.

Now, using the definition of the $g^{(t)}$ functions, and by the regret bound for the EG algorithm, we have that for any fixed distribution $\alpha \in \Delta_N$,

$$\sum_{t=1}^{T} f^{(t)}(\phi_\alpha(\mathbf{x}^{(t)})) - \sum_{t=1}^{T} f^{(t)}(\phi_{\alpha^{(t)}}(\mathbf{x}^{(t)})) \;=\; \sum_{t=1}^{T} g^{(t)}(\alpha) - \sum_{t=1}^{T} g^{(t)}(\alpha^{(t)}) \;\leq\; O(\sqrt{\log(N)T}). \quad (3)$$

Since $\|\nabla f^{(t)}\| \leq 1$,

$$f^{(t)}(\phi_{\alpha^{(t)}}(\mathbf{x}^{(t)})) - f^{(t)}(\mathbf{x}^{(t)}) \;\leq\; \|\phi_{\alpha^{(t)}}(\mathbf{x}^{(t)}) - \mathbf{x}^{(t)}\| \;\leq\; \frac{1}{\sqrt{t}}. \quad (4)$$

Summing (4) from $t = 1$ to $T$, and adding to (3), we get that for any distribution $\alpha$ over $\Phi$,

$$\sum_{t=1}^{T} f^{(t)}(\phi_\alpha(\mathbf{x}^{(t)})) - \sum_{t} f^{(t)}(\mathbf{x}^{(t)}) \;\leq\; O(\sqrt{\log(N)T}) + \sum_{t=1}^{T} \frac{1}{\sqrt{t}} \;=\; O(\sqrt{\log(N)T}).$$

In particular, by concentrating $\alpha$ on any given $\phi_i$, the above inequality implies that $\sum_{t=1}^{T} f^{(t)}(\phi_i(\mathbf{x}^{(t)})) - \sum_{t=1}^{T} f^{(t)}(\mathbf{x}^{(t)}) \leq O(\sqrt{\log(N)T})$, and thus we have a no $\Phi$-regret algorithm. $\square$

## Footnotes

[1]It is highly plausible that the results in this paper extend to the case where $\Phi$ is infinite – indeed, our results hold for any set of mappings $\Phi$ which is obtained by taking all convex combinations of finitely many mappings – but we restrict to finite $\Phi$ in this paper for simplicity.

[2] In the full version of the paper, we improve the regret bound to $O(\log T)$ under some stronger concavity assumptions on the payoff functions.

## References

[1] S. Arora, E. Hazan, and S. Kale. The multiplicative weights update method: a meta algorithm and applications. *Manuscript*, 2005.

[2] A. Blum and Y. Mansour. From external to internal regret. In *COLT*, pages 621–636, 2005.

[3] X. Chen and X. Deng. Settling the complexity of two-player nash equilibrium. In *47th FOCS*, pages 261–272, 2006.

[4] X. Chen, X. Deng, and S-H. Teng. Computing nash equilibria: Approximation and smoothed complexity. *focs*, 0:603–612, 2006.

[5] T. Cover. Universal portfolios. *Math. Finance*, 1:1–19, 1991.

[6] C. Daskalakis, P. W. Goldberg, and C. H. Papadimitriou. The complexity of computing a nash equilibrium. In *38th STOC*, pages 71–78, 2006.

[7] Y. Freund and R. E. Schapire. Adaptive game playing using multiplicative weights. *Games and Economic Behavior*, 29:79–103, 1999.

[8] G. Gordon, A. Greenwald, C. Marks, and M. Zinkevich. No-regret learning in convex games. *Brown University Tech Report CS-07-10*, 2007.

[9] A. Greenwald and A. Jafari. A general class of no-regret learning algorithms and game-theoretic equilibria, 2003.

[10] J. Hannan. Approximation to bayes risk in repeated play. *In M. Dresher, A. W. Tucker, and P. Wolfe, editors, Contributions to the Theory of Games, volume III*, pages 97–139, 1957.

[11] S. Hart and A. Mas-Colell. A simple adaptive procedure leading to correlated equilibrium. *Econometrica*, 68(5):1127–1150, 2000.

[12] S. Hart and D. Schmeidler. Existence of correlated equilibria. *Mathematics of Operations Research*, 14(1):18–25, 1989.

[13] E. Hazan, A. Kalai, S. Kale, and A. Agarwal. Logarithmic regret algorithms for online convex optimization. *In 19'th COLT*, 2006.

[14] J. Kivinen and M. K. Warmuth. Exponentiated gradient versus gradient descent for linear predictors. *Inf. Comput.*, 132(1):1–63, 1997.

[15] C. H. Papadimitriou. On the complexity of the parity argument and other inefficient proofs of existence. *J. Comput. Syst. Sci.*, 48(3):498–532, 1994.

[16] G. Stoltz and G. Lugosi. Internal regret in on-line portfolio selection. *Machine Learning*, 59:125–159, 2005.

[17] G. Stoltz and G. Lugosi. Learning correlated equilibria in games with compact sets of strategies. *Games and Economic Behavior*, 59:187–208, 2007.

[18] M. Zinkevich. Online convex programming and generalized infinitesimal gradient ascent. In *20th ICML*, pages 928–936, 2003.

